# Probabilistic computation in spiking populations

**Richard S. Zemel**
Dept. of Comp. Sci.
Univ. of Toronto

**Quentin J. M. Huys**
Gatsby CNU
UCL

**Rama Natarajan**
Dept. of Comp. Sci.
Univ. of Toronto

**Peter Dayan**
Gatsby CNU
UCL

## Abstract

As animals interact with their environments, they must constantly update estimates about their states. Bayesian models combine prior probabilities, a dynamical model and sensory evidence to update estimates optimally. These models are consistent with the results of many diverse psychophysical studies. However, little is known about the neural representation and manipulation of such Bayesian information, particularly in populations of spiking neurons. We consider this issue, suggesting a model based on standard neural architecture and activations. We illustrate the approach on a simple random walk example, and apply it to a sensorimotor integration task that provides a particularly compelling example of dynamic probabilistic computation.

Bayesian models have been used to explain a gamut of experimental results in tasks which require estimates to be derived from multiple sensory cues. These include a wide range of psychophysical studies of perception;[13] motor action;[7] and decision-making.[3,5] Central to Bayesian inference is that computations are sensitive to *uncertainties* about afferent and efferent quantities, arising from ignorance, noise, or inherent ambiguity (e.g., the aperture problem), and that these uncertainties change over time as information accumulates and dissipates. Understanding how neurons represent and manipulate uncertain quantities is therefore key to understanding the neural instantiation of these Bayesian inferences.

Most previous work on representing probabilistic inference in neural populations has focused on the representation of static information.[1,12,15] These encompass various strategies for encoding and decoding uncertain quantities, but do not readily generalize to real-world dynamic information processing tasks, particularly the most interesting cases with stimuli changing over the same timescale as spiking itself.[11] Notable exceptions are the recent, seminal, but, as we argue, representationally restricted, models proposed by Gold and Shadlen,[5] Rao,[10] and Deneve.[4]

In this paper, we first show how probabilistic information varying over time can be represented in a spiking population code. Second, we present a method for producing spiking codes that facilitate further processing of the probabilistic information. Finally, we show the utility of this method by applying it to a temporal sensorimotor integration task.

## 1  TRAJECTORY ENCODING AND DECODING

We assume that population spikes $R(t)$ arise stochastically in relation to the trajectory $X(t)$ of an underlying (but hidden) variable. We use $\mathbf{R}_T$ and $\mathbf{X}_T$ for the whole trajectory and

spike trains respectively from time $0$ to $T$. The spikes $\mathbf{R}_T$ constitute the observations and are assumed to be probabilistically related to the signal by a tuning function $f(X, \theta_i)$:

$$P(R(i,T)|X(T)) \propto f(X, \theta_i) \tag{1}$$

for the spike train of the $i$th neuron, with parameters $\theta_i$. Therefore, via standard Bayesian inference, $\mathbf{R}_T$ determines a distribution over the hidden variable at time $T$, $P(X(T)|\mathbf{R}_T)$.

We first consider a version of the dynamics and input coding that permits an analytical examination of the impact of spikes. Let $X(t)$ follow a stationary Gaussian process such that the joint distribution $P(X(t_1), X(t_2), \dots, X(t_m))$ is Gaussian for any finite collection of times, with a covariance matrix which depends on time differences: $\mathcal{C}_{tt'} = c(|t - t'|)$. Function $c(|\Delta t|)$ controls the smoothness of the resulting random walks. Then,

$$P(X(T)|\mathbf{R}_T) \propto p(X(T)) \int_{\mathbf{X}(T)} d\mathbf{X}(T) P(\mathbf{R}_T|\mathbf{X}(T)) P(\mathbf{X}(T)|X(T)) \tag{2}$$

where $P(\mathbf{X}(T)|X(T))$ is the distribution over the whole trajectory $\mathbf{X}(T)$ conditional on the value of $X(T)$ at its end point. If $\mathbf{R}_T$ are a set of conditionally independent inhomogeneous Poisson processes, we have

$$P(\mathbf{R}_T|\mathbf{X}(T)) \propto \prod_{i\tau} f(X(t_{i\tau}), \theta_i) \exp\left(-\sum_i \int_\tau d\tau\; f(X(\tau), \theta_i)\right), \tag{3}$$

where $t_{i\tau} \forall \tau$ are the spike times $\tau$ of neuron $i$ in $\mathbf{R}_T$. Let $\boldsymbol{\chi} = [X(t_{i\tau})]$ be the vector of stimulus positions at the times at which we observed a spike and $\boldsymbol{\Theta} = [\theta(t_{i\tau})]$ be the vector of spike positions. If the tuning functions are Gaussian $f(X, \theta_i) \propto \exp(-(X - \theta_i)^2/2\sigma^2)$ and sufficiently dense that $\sum_i \int_\tau d\tau\; f(X, \theta_i)$ is independent of $X$ (a standard assumption in population coding), then $P(\mathbf{R}_T|\mathbf{X}(T)) \propto \exp(-\|\boldsymbol{\chi} - \boldsymbol{\Theta}\|^2/2\sigma^2)$ and in Equation 2, we can marginalize out $\mathbf{X}(T)$ except at the spike times $t_{i\tau}$:

$$P(X(T)|\mathbf{R}_T) \propto p(X(T)) \int_{\boldsymbol{\chi}} d\boldsymbol{\chi} \exp\left(-[\boldsymbol{\chi}, X(T)]^\mathsf{T} \tfrac{\mathcal{C}^{-1}}{2}[\boldsymbol{\chi}, X(T)] - \tfrac{\|\boldsymbol{\chi} - \boldsymbol{\Theta}\|^2}{2\sigma^2}\right) \tag{4}$$

and $\mathcal{C}$ is the block covariance matrix between $X(t_{i\tau}), x(T)$ at the spike times $[t_{t\tau}]$ and the final time $T$. This Gaussian integral has $P(X(T)|\mathbf{R}_T) \sim \mathcal{N}(\mu(T), \nu(T))$, with

$$\mu(T) = \mathcal{C}_{Tt}(\mathcal{C}_{tt} + \mathbf{I}\sigma^2)^{-1}\boldsymbol{\Theta} = \mathbf{k}\boldsymbol{\Theta} \qquad \nu(T) = \mathcal{C}_{TT} - \mathbf{k}\mathcal{C}_{tT} \tag{5}$$

$\mathcal{C}_{TT}$ is the $T, T$th element of the covariance matrix and $\mathcal{C}_{Tt}$ is similarly a row vector. The dependence in $\mu$ on past spike times is specified chiefly by the inverse covariance matrix, and acts as an effective kernel ($\mathbf{k}$). This kernel is *not* stationary, since it depends on factors such as the local density of spiking in the spike train $\mathbf{R}_T$.

For example, consider where $X(t)$ evolves according to a diffusion process with drift:

$$dX = -\alpha X dt + \sigma_\epsilon dN(t) \tag{6}$$

where $\alpha$ prevents it from wandering too far, $N(t)$ is white Gaussian noise with mean zero and $\sigma_\epsilon^2$ variance. Figure 1A shows sample kernels for this process.

Inspection of Figure 1A reveals some important traits. First, the monotonically decreasing kernel magnitude as the time span between the spike and the current time $T$ grows matches the intuition that recent spikes play a more significant role in determining the posterior over $X(T)$. Second, the kernel is nearly exponential, with a time constant that depends on the time constant of the covariance function and the density of the spikes; two settings of these parameters produced the two groupings of kernels in the figure. Finally, the fully adaptive kernel $\mathbf{k}$ can be locally well approximated by a *metronomic* kernel $\mathbf{k}^{<\mathbf{R}>}$ (shown in red in Figure 1A) that assumes regular spiking. This takes advantage of the general fact, indicated by the grouping of kernels, that the kernel depends weakly on the actual spike pattern, but strongly on the average rate. The merits of the metronomic kernel are that it is stationary and only depends on a single mean rate rather than the full spike train $\mathbf{R}_T$. It also justifies

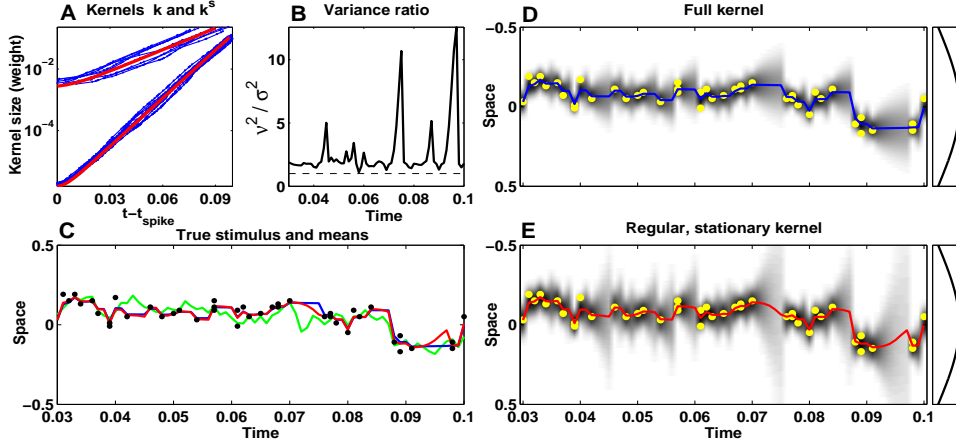

Figure 1: Exact and approximate spike decoding with the Gaussian process prior. Spikes are shown in yellow, the true stimulus in green, and $P(X(T)|\mathbf{R}_T)$ in gray. **Blue**: exact inference with nonstationary and **red**: approximate inference with regular spiking. **A** Kernel samples for a diffusion process as defined by equations 5, 6. **B, C**: Mean and variance of the inference. **D**: Exact inference with full kernel **k** and **E:** approximation based on metronomic kernel $\mathbf{k}^{<\mathbf{R}>}$. (Equation 7).

the form of decoder used for the network model in the next section.[6] Figure 1D shows an example of how well Equation 5 specifies a distribution over $X(t)$ through very few spikes.

Finally, 1E shows a factorized approximation with the stationary kernel similar to that used by Hinton and Brown[6] and in our recurrent network:

$$\hat{P}(X(t)|\mathbf{R}(t)) \propto \prod_i f(X, \theta_i)^{\sum_{j=0}^t k_j^s t_{ij}} = \exp(-E(X(t), \mathbf{R}(t), t)), \qquad (7)$$

By design, the mean is captured very well, but not the variance, which in this example grows too rapidly for long interspike intervals (Figure 1B, C). Using a slower kernel improves performance on the variance, but at the expense of the mean. We thus turn to the network model with recurrent connections that are available to reinstate the spike-conditional characteristics of the full kernel.

## 2 NETWORK MODEL FORMULATION

Above we considered how population spikes $\mathbf{R}_T$ specify a distribution over $X(T)$. We now extend this to consider how interconnected populations of neurons can specify distributions over time-varying variables. We frame the problem and our approach in terms of a two-level network, connecting one population of neurons to another; this construction is intended to apply to any level of processing. The network maps input population spikes $R(t)$ to output population spikes $S(t)$, where input and output evolve over time. As with the input spikes, $\mathbf{S}_T$ indicates the output spike trains from time $0$ to $T$, and these output spikes are assumed to determine a distribution over a related hidden variable.

For the recurrent and feedforward computation in the network, we start with the deceptively simple goal[9] of producing output spikes in such a way that the distribution $Q(X(T)|\mathbf{S}_T)$ they imply over the *same* hidden variable $X(T)$ as the input, faithfully matches $P(X(T)|\mathbf{R}_T)$. This might seem a strange goal, since one could surely just listen to the input spikes. However, in order for the output spikes to track the hidden variable, the dynamics of the interactions between the neurons must explicitly capture the dynamics

of the process $\mathbf{X}(T)$. Once this 'identity mapping' problem has been solved, more general, complex computations can be performed with ease. We illustrate this on a multisensory integration task, tracking a hidden variable that depends on multiple sensory cues.

The aim of the recurrent network is to take the spikes $R(t)$ as inputs, and produce output spikes that capture the probabilistic dynamics. We proceed in two steps. We first consider the probabilistic decoding process which turns $\mathbf{S}_T$ into $Q(X(t)|\mathbf{S}_T)$. Then we discuss the recurrent and feedforward processing that produce appropriate $\mathbf{S}_T$ given this decoder. Note that this decoding process is not required for the network processing; it instead provides a computational objective for the spiking dynamics in the system.

We use a simple log-linear decoder based on a spatiotemporal kernel:[6]

$$Q(X(T)|\mathbf{S}_T) \propto \exp(-E(X(T), \mathbf{S}_T, T)) \text{, where} \tag{8}$$

$$E(X, \mathbf{S}_T, T) = \sum_j \sum_{\tau=0}^T S(j, T-\tau)\phi_j(X, \tau) \tag{9}$$

is an energy function, and the spatiotemporal kernels are assumed separable: $\phi_j(X, \tau) = g_j(X)\psi(\tau)$. The spatial kernel $g_j(X)$ is related to the receptive field $f(X, \theta_j)$ of neuron $j$ and the temporal kernel $\phi_j(X, \tau)$ to $\mathbf{k}^{<\mathbf{R}_T>}$.

The dynamics of processing in the network follows a standard recurrent neural architecture for modeling cortical responses, in the case that network inputs $R(t)$ and outputs $S(t)$ are spikes. The effect of a spike on other neurons in the network is assumed to have some simple temporal dynamics, described here again by the temporal kernel $\psi(\tau)$:

$$r_i(t) = \sum_{\tau=0}^T R(i, T-\tau)\psi(\tau) \quad s_j(t) = \sum_{\tau=0}^T S(j, T-\tau)\psi(\tau)$$

where $T$ is the extent of the kernel. The response of an output neuron is governed by a stochastic spiking rule, where the probability that neuron $j$ spikes at time $t$ is given by:

$$P(S(j, t) = 1) = \sigma(u_j(t)) = \sigma\left(\sum_i w_{ij}r_i(t) + \sum_k v_{kj}s_k(t-1)\right) \tag{10}$$

where $\sigma()$ is the logistic function, and $\mathbf{W}$ and $\mathbf{V}$ are the feedforward and recurrent weights. If $\psi(\tau) = \exp(-\kappa\tau)$, then $u_j(T) = \psi(0)(W_j \cdot R(T) + V_j \cdot S(T)) + \psi(1)u_j(T-1)$; this corresponds to a discretization of the standard dynamics for the membrane potential of a leaky integrate-and-fire neuron: $\tau\frac{du_j}{dt} = -\gamma u_j + \mathbf{W}R + \mathbf{V}S$, where the leak $\gamma$ is determined by the temporal kernel.

The task of the network is to make $Q(X(T)|\mathbf{S}_T)$ of Equation 8 match $P(X(T)|\mathbf{R}_T)$ coming from one of the two models above (exact dynamic or approximate stationary kernel). We measure the discrepancy using the Kullback-Leibler (KL) divergence:

$$\mathcal{J} = \sum_t KL\left[P(X(T)|\mathbf{R}_T)||Q(X(T)|\mathbf{S}_T)\right] \tag{11}$$

and, as a proof of principle in the experiments below, find optimal $\mathbf{W}$ and $\mathbf{V}$ by minimizing the KL divergence $\mathcal{J}$ using back-propagation through time (BPTT). In order to implement this in the most straightforward way, we convert the stochastic spiking rule (Equation 10) to a deterministic rule via the mean-field assumption: $S_j(t) = \sigma\left(\sum_i w_{ij}r_i(t) + \sum_k v_{kj}s_k(t-1)\right)$. The gradients are tedious, but can be neatly expressed in a temporally recursive form. Note that our current focus in the system is on the representational capability of the system, rather than its learning. Our results establish that the system can faithfully represent the posterior distribution. We return to the issue of more plausible learning rules below.

The resulting network can be seen as a dynamic spiking analogue of the recurrent network scheme of Pouget et al.:[9] both methods formulate feedforward and recurrent connections so that a simple decoding of the output can match optimal but complex decoding applied to the inputs. A further advantage of the scheme proposed here is that it facilitates downstream processing of the probabilistic information, as the objective encourages the formation of distributions at the output that factorize across the units.

# 3 RELATED MODELS

Ideas about the representation of probabilistic information in spiking neurons are in vogue. One treatment considers Poisson spiking in populations with regular tuning functions, assuming that stimuli change slowly compared with the inter-spike intervals.[8] This leads to a Kalman filter account with much formal similarity to the models of $P(X(T)|\mathbf{R}_T)$. However, because of the slow timescale, recurrent dynamics can be allowed to settle to an underlying attractor. In another approach, the spiking activity of either a single neuron[4] or a pair of neurons[5] is considered as reporting (logarithmic) probabilistic information about an underlying binary hypothesis. A third treatment proposes that a population of neurons directly represents the (logarithmic) probability over the state of a hidden Markov model.[10]

Our method is closely related to the latter two models. Like Deneve's[4] we consider the transformation of input spikes to output spikes with a fixed assumed decoding scheme so that the dynamics of an underlying process is captured. Our decoding mechanism produces something like the predictive coding apparent in Deneve's scheme, except that here, a neuron may not need to spike not only if it itself has recently spiked and thereby conveyed the appropriate information; but also if one of its population neighbors has recently spiked. This is explicitly captured by the recurrent interactions among the population. Our scheme also resembles Rao's[10] approach in that it involves population codes. Our representational scheme is more general, however, in that the spatiotemporal decoder defines the relationship between output spikes and $Q(X(T)|\mathbf{S}_T)$, whereas his method assumes a direct encoding, with each output neuron's activity proportional to $\log Q(X(T)|\mathbf{S}_T)$. Our decoder can produce such a direct encoding if the spatial and temporal kernels are delta functions, but other kernels permit coordination amongst the population to take into account temporal effects, and to produce higher fidelity in the output distribution.

# 4 EXPERIMENTS

**1. Random walk.** We describe two experiments. For ease of presentation and comparison, these simulations treat $X(t)$ as a discrete variable, so that the encoding model is a hidden Markov model (HMM) rather than the Gaussian process defined above. The first is a random walk, as in Equation 6 and Figure 1, which allows us to make comparisons with the exact statistics. In a discrete setting, the walk parameters $\alpha$ and $\sigma_\epsilon$ determine the entries in the transition matrix of the corresponding HMM; in a continuous one, the covariance function $\mathcal{C}$ of the Gaussian process. Input spikes are generated according to Gaussian tuning functions (Equation 1). In the recurrent network model, the spatiotemporal kernels are fixed: the spatial kernels are based on the regular locations of the output units $j$, $g_j(X) = |X - X_j|^2/(1 + |X - X_j|^2)$ and the temporal kernel is $\psi(\tau) = \exp(-\kappa\tau)$, where $\kappa = 2$ is set to match the walk dynamics. In the following simulations, the network contained 20 inputs, 60 states, and 20 outputs.

Results on two walk trajectories with different dynamics are shown in Figure 2. The network is trained on walks with parameters ($\alpha = 0.2$, $\sigma_\epsilon = 2$) that force the state to move to and remain near the center. Figures 2A & B show that in intervals without input spikes, the inferred mean quickly shifts towards the center and remains there until evidence is received in the form of input spikes. The feedforward weights (Fig. 2F) show strong connections between an input unit and its corresponding output, while the learned recurrent weights (Fig. 2E) reflect the transition probabilities: units coding for extreme values have strong connections to those nearer the center, and units with preferred values near the center have strong self-connections. Fig. 2C&D) shows the results of testing this trained network on walks with different dynamics ($\alpha = 0.8$, $\sigma_\epsilon = 7$). The resulting mismatch between the mean approximated trajectory (yellow line) and true stimulus (dashed line) (Fig. 2D), and the variance (Fig. 2H), shows that the learned weights capture the input dynamics.

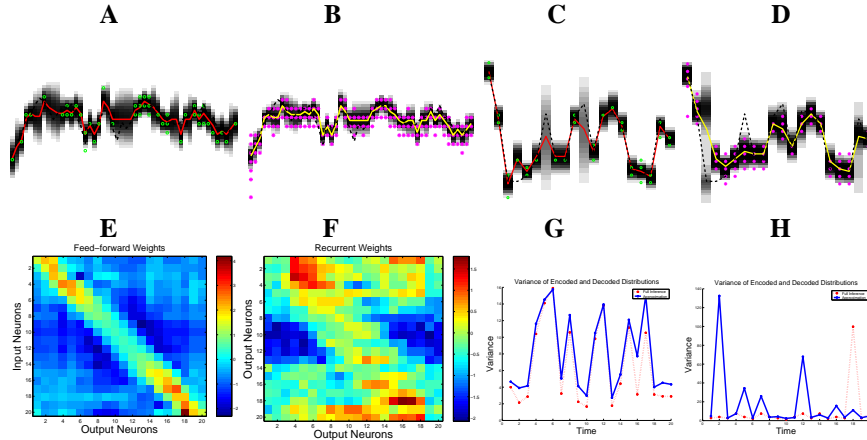

Figure 2: Comparison between full inference using hidden Markov model and approximation using network model. Top Row: Full Inference (A,C) and approximation (B,D) results from two walks. Input spikes ($\mathbf{R}_T$) are shown as green circles; output spikes ($\mathbf{S}_T > .9$) as magenta stars; true stimulus as dashed line; mean inferred trajectory as red line; mean approximated trajectory as yellow line; distributions $P(X(t)|\mathbf{R}_T)$ and $Q(X(t)|\mathbf{S}_T)$ at each timestep in gray. Bottom Row: Feedfoward (E); recurrent weights (F); variance of exact and approximate inference from walks 1 (G) and 2 (H).

**2. Sensorimotor task.** We next applied our framework to a recent experiment on probabilistic computation during sensorimotor processing.[7] Here, human subjects tried to move a cursor on a display to a target by moving a (hidden) finger. The cursor was shown before the start of the movement, it was then hidden, except for one point of blurry visual feedback in the middle of the movement (with variances $0 = \sigma_0 < \sigma_L < \sigma_M < \sigma_\infty = \infty$). Unbeknownst to them, on the onset of movement, the cursor was displaced by $\Delta X$, drawn from a prior distribution $P(\Delta X)$. The subjects must estimate $\Delta X$ in order to compensate for the displacement and land the cursor on the target. The key result is that subjects learned and used the prior information $P(\Delta X)$, and indeed integrated it with the visual information in a way that was appropriately sensitive to the amount of blur (figure 3A). The authors showed that a simple Bayesian model could account for these data.

We model a population representation of the 2D cursor position $X(t)$ on the screen. Two spiking input codes—from vision ($\mathbf{R}_T{}^v$) and proprioception ($\mathbf{R}_T{}^p$), present also in the absence of visual feedback—are mapped into an output code $\mathbf{S}_T$ representing $P(X(t)|\mathbf{R}_T{}^v, \mathbf{R}_T{}^p)$. This is a neural instantiation of Bayesian cue combination, and also an extension of the previous model to the dynamic case.

The first experiment involved a Gaussian prior distribution: $P(\Delta X) \sim \mathcal{N}(1, .5)$. During initial experience subjects learn about this prior from trajectories; we determine the parameters of the HMM. We use BPTT to learn feedforward weights for the input spikes from the different modalities, and recurrent weights between the output units. The input population had 64 units per modality, while the state space and output population each had 100 units. Input spikes were Poisson based on tuning functions centered on a grid within the 2D space; spatiotemporal kernels were based on the (gridded) output units $j$. The model was tested in conditions directly matching the experiment, with the cursor and finger moving along a straight line trajectory from the current model's estimate of the cursor position, $< X(t) >_{Q(X(t)|\mathbf{S}_T)}$, to the target location. The model captures the main effects of the experiment (see Figure 3) with respect to visual blur. The prior was ignored when the sensory evidence was precise ($\sigma_0$), it dominated on trials without feedback ($\sigma_\infty$), and the two

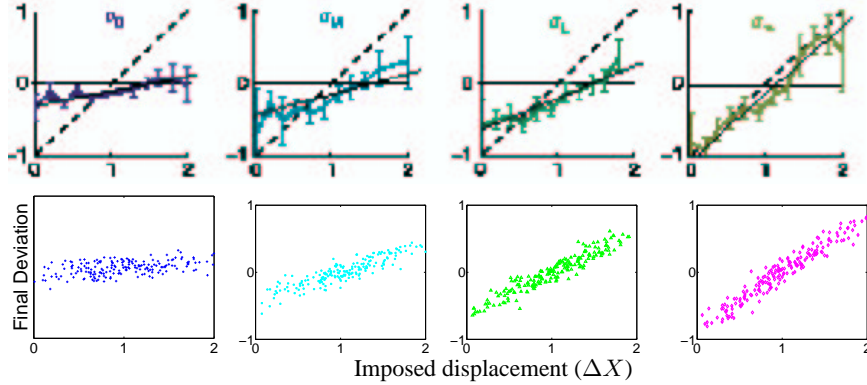

Figure 3: (a) Results for a typical subject from the first Körding-Wolpert experiment,[7] for different degrees of blur in the visual feedback ($\sigma_{\{0,M,L,\infty\}}$). The average lateral displacement of the cursor from the target location at the trial end, as a function of the imposed displacement of the cursor from the finger location ($\Delta X$), which is drawn from $\mathcal{N}(1, .5)$. (b) Model results under the same testing conditions. See text for model details.

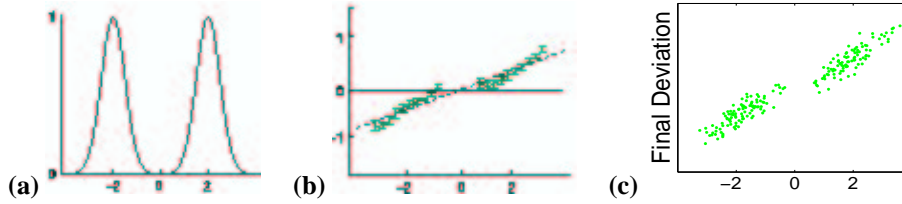

Figure 4: (a) Bimodal prior $P(\Delta X) \sim \mathcal{N}(\pm 2, 0.5)$ for cursor displacement in second Körding-Wolpert experiment.[7] (b) Results from human subjects. (c) Model results.

factors combined on intermediate degrees of blurriness.

In the second experiment the prior was bimodal (Fig. 4A) and feedback was blurred ($\sigma_L$). For this prior, the final cursor location should be based on the more prevalent displacements, so responses based on optimal inference should be non-linear. may modify the posterior estimate of cursor location. Finally, its 2DIndeed, this is the case (Fig. 4B;C). Intuitively, the blurry visual feedback inadequately defines the true finger position, and thus the posterior $P(X(t)|\mathbf{R}_T)$ is influenced by the learned bimodal prior; the network model accurately reconstructs this optimal posterior.

Our model generalizes the simple Bayesian account, and suggests new avenues for predictions. The dynamic nature of the system permits modeling the integration of several visual cues during the trial, as well as differential effects of the timing of visual feedback. The integration of cues in our model also allows it to capture interactions between them. Finally, its 2D nature allows our system to model other aspects of combining visual and proprioceptive cues, such as their varying and contrasting degrees of certainty across space.[14]

## 5 DISCUSSION

We proposed a spiking population coding scheme for representing and propagating uncertainty which operates at the fine-grained timescale of individual inter-spike intervals. We motivated the key spatio-temporal spike kernels in the model from analytical results in a

Gaussian process, and suggested two approximations to the exact decoding provided by these adaptive spatiotemporal kernels. The first is a regular stationary kernel while the second is a recurrent network model. We showed how gradient descent can set model parameters to match the requirements on the output distribution and capture the dynamics underlying a hidden variable. This is a dynamic and spiking extension of DPC,[15] and a population extension of Deneve.[4] We showed its proficiency by comparison with exact inference in a random walk, and a neural model that does not use a population code.

The most important direction concerns biologically plausible learning in the full spiking form of the model. One possibility is to view spikes as a primitive action chosen by a neuron. In this case, we can use the analog of direct policy methods in partially observable Markov decision processes,[2] with faithful tracking of $X(t)$ leading to reward. It is also possible that simpler, Hebbian rules will suffice. A second future direction concerns inference of one variable from another using our spiking population code model. This problem involves marginalizing over intermediate variables, which is difficult in direct representations of distributions over these variables, due to approximating logs of sums with sums of logs;[10] we are investigating how well our scheme can approximate this computation.

We applied the model to a challenging sensorimotor integration task which has been used to demonstrate Bayesian inference. Since it offers a dynamic account, we can make a number of predictions about the consequences of variations to the experiment. Most interesting would be the case in which a bimodal likelihood is combined with a unimodal (or bimodal) prior (rather than vice-versa), or indeed two instances of visual feedback during the task.

**Acknowledgements**

We thank Sophie Deneve and Jon Pillow for helpful discussions. RZ & RN funded by NSERC, CIHR NET program; PD & QH by Gatsby Charitable Fdtn., BIBA consortium, UCL MB/PhD program.

# References

[1] Anderson C.H. & Van Essen D.C. (1994). Neurobiological computational systems. *In: Computational Intelligence: Imitating Life, Zurada, Marks, Robinson (ed.)*, IEEE Press, 213-222.

[2] Baxter, J. & Bartlett, P. (2001). Infinite-horizon policy-gradient estimation. *JAIR*, 319 - 350.

[3] Carpenter, R. H. S. & Williams, M. L. L. (1995). Neural computation of log likelihood in the control of saccadic eye movements. *Nature*, 377: 59-62.

[4] Deneve, S. (2004). Bayesian inference in spiking neurons. NIPS-17.

[5] Gold, JI & Shadlen, MN (2001). Neural computations that underlie decisions about sensory stimuli. *Trends in Cognitive Sciences* **5**:10-16.

[6] Hinton, GE & Brown, AD (2000) Spiking Boltzmann machines. NIPS-12: 122-129.

[7] Körding, KP & Wolpert, D (2004) Bayesian integration in sensorimotor learning. *Nature* **427**:244-247.

[8] Latham, P., Deneve, S., & Pouget, A., (2004). Optimal computation with attractor networks. *J Physiology, Paris*.

[9] Pouget, A., Zhang, K, Deneve, S, & Latham, P. (1998) Statistically efficient estimation using population codes. *NeuralComputation,* **10**: 373-401.

[10] Rao, R. (2004). Bayesian computation in recurrent neural circuits. *Neural Computation*, 16(1).

[11] Rieke, F, Warland, D, de Ruyter v. Steveninck, & Bialek, W. (1999). *Spikes*. MIT Press.

[12] Sahani, M & Dayan, P (2003) Doubly distributional population codes: Simultaneous representation of uncertainty and multiplicity. *Neural Computation* **15**.

[13] Saunders, J.A. & Knill, D.C. (2001). Perception of 3D surface orientation from skew symmetry. *Vision Research*, 41 (24) 3163 - 3183.

[14] Van Beers, R. J., Sittig, A.C., & Denier, J.J. (1999). Integration of propriocetive and visual position-information *J Neurophysiol*, 81: 1355-1364.

[15] Zemel, R.S., Dayan, P. & Pouget A. (1998). Probabilistic interpretation of population codes. *Neural Computation,* **10**, 403-430.
